# Global Optimisation of Neural Network Models Via Sequential Sampling

**João FG de Freitas**
Cambridge University
Engineering Department
Cambridge CB2 1PZ England
jfgf@eng.cam.ac.uk
[Corresponding author]

**Arnaud Doucet**
Cambridge University
Engineering Department
Cambridge CB2 1PZ England
ad2@eng.cam.ac.uk

**Mahesan Niranjan**
Cambridge University
Engineering Department
Cambridge CB2 1PZ England
niranjan@eng.cam.ac.uk

**Andrew H Gee**
Cambridge University
Engineering Department
Cambridge CB2 1PZ England
ahg@eng.cam.ac.uk

## Abstract

We propose a novel strategy for training neural networks using sequential sampling-importance resampling algorithms. This global optimisation strategy allows us to learn the probability distribution of the network weights in a sequential framework. It is well suited to applications involving on-line, nonlinear, non-Gaussian or non-stationary signal processing.

## 1  INTRODUCTION

This paper addresses sequential training of neural networks using powerful sampling techniques. Sequential techniques are important in many applications of neural networks involving real-time signal processing, where data arrival is inherently sequential. Furthermore, one might wish to adopt a sequential training strategy to deal with non-stationarity in signals, so that information from the recent past is lent more credence than information from the distant past. One way to sequentially estimate neural network models is to use a state space formulation and the extended Kalman filter (Singhal and Wu 1988, de Freitas, Niranjan and Gee 1998). This involves local linearisation of the output equation, which can be easily performed, since we only need the derivatives of the output with respect to the unknown parameters. This approach has been employed by several authors, including ourselves.

However, local linearisation leading to the EKF algorithm is a gross simplification of the probability densities involved. Nonlinearity of the output model induces multi-modality of the resulting distributions. Gaussian approximation of these densities will loose important details. The approach we adopt in this paper is one of sampling. In particular, we discuss the use of 'sampling-importance resampling' and 'sequential importance sampling' algorithms, also known as particle filters (Gordon, Salmond and Smith 1993, Pitt and Shephard 1997), to train multi-layer neural networks.

## 2   STATE SPACE NEURAL NETWORK MODELLING

We start from a state space representation to model the neural network's evolution in time. A transition equation describes the evolution of the network weights, while a measurements equation describes the nonlinear relation between the inputs and outputs of a particular physical process, as follows:

$$\mathbf{w}_{k+1} = \mathbf{w}_k + \mathbf{d}_k \tag{1}$$
$$\mathbf{y}_k = \mathbf{g}(\mathbf{w}_k, \mathbf{x}_k) + \mathbf{v}_k \tag{2}$$

where ($\mathbf{y}_k \in \Re^o$) denotes the output measurements, ($\mathbf{x}_k \in \Re^d$) the input measurements and ($\mathbf{w}_k \in \Re^m$) the neural network weights. The measurements nonlinear mapping $\mathbf{g}(.)$ is approximated by a multi-layer perceptron (MLP). The measurements are assumed to be corrupted by noise $\mathbf{v}_k$. In the sequential Monte Carlo framework, the probability distribution of the noise is specified by the user. In our examples we shall choose a zero mean Gaussian distribution with covariance $R$. The measurement noise is assumed to be uncorrelated with the network weights and initial conditions.

We model the evolution of the network weights by assuming that they depend on the previous value $\mathbf{w}_k$ and a stochastic component $\mathbf{d}_k$. The process noise $\mathbf{d}_k$ may represent our uncertainty in how the parameters evolve, modelling errors or unknown inputs. We assume the process noise to be a zero mean Gaussian process with covariance $Q$, however other distributions can also be adopted. This choice of distributions for the network weights requires further research. The process noise is also assumed to be uncorrelated with the network weights.

The posterior density $p(W_k|Y_k)$, where $Y_k = \{\mathbf{y}_1, \mathbf{y}_2, \cdots, \mathbf{y}_k\}$ and $W_k = \{\mathbf{w}_1, \mathbf{w}_2, \cdots, \mathbf{w}_k\}$, constitutes the complete solution to the sequential estimation problem. In many applications, such as tracking, it is of interest to estimate one of its marginals, namely the filtering density $p(\mathbf{w}_k|Y_k)$. By computing the filtering density recursively, we do not need to keep track of the complete history of the weights. Thus, from a storage point of view, the filtering density turns out to be more parsimonious than the full posterior density function. If we know the filtering density of the network weights, we can easily derive various estimates of the network weights, including centroids, modes, medians and confidence intervals.

## 3   SEQUENTIAL IMPORTANCE SAMPLING

In the sequential importance sampling optimisation framework, a set of representative samples is used to describe the posterior density function of the network parameters. Each sample consists of a complete set of network parameters. More specifically, we make use of the following Monte Carlo approximation:

$$\hat{p}(W_k|Y_k) = \frac{1}{N} \sum_{i=1}^{N} \delta(W_k - W_k^{(i)})$$

where $W_k^{(i)}$ represents the samples used to describe the posterior density and $\delta(.)$ denotes the Dirac delta function. Consequently, any expectations of the form:

$$\mathbf{E}[f_k(W_k)] = \int f_k(W_k)\mathrm{p}(W_k|Y_k)dW_k$$

may be approximated by the following estimate:

$$\mathbf{E}[f_k(W_k)] \approx \frac{1}{N}\sum_{i=1}^{N} f_k(W_k^{(i)})$$

where the samples $W_k^{(i)}$ are drawn from the posterior density function. Typically, one cannot draw samples directly from the posterior density. Yet, if we can draw samples from a proposal density function $\pi(W_k|Y_k)$, we can transform the expectation under $\mathrm{p}(W_k|Y_k)$ to an expectation under $\pi(W_k|Y_k)$ as follows:

$$
\begin{aligned}
\mathbf{E}[f_k(W_k)] &= \int f_k(W_k)\frac{\mathrm{p}(W_k|Y_k)}{\pi(W_k|Y_k)}\pi(W_k|Y_k)dW_k \\
&= \frac{\int f_k(W_k)q_k(W_k)\pi(W_k|Y_k)dW_k}{\int q_k(W_k)\pi(W_k|Y_k)dW_k} \\
&= \frac{\mathbf{E}_\pi[q_k(W_k)f_k(W_k)]}{\mathbf{E}_\pi[q_k(W_k)]}
\end{aligned}
$$

where the variables $q_k(W_k)$ are known as the unnormalised importance ratios:

$$q_k = \frac{\mathrm{p}(Y_k|W_k)\mathrm{p}(W_k)}{\pi(W_k|Y_k)} \tag{3}$$

Hence, by drawing samples from the proposal function $\pi(.)$, we can approximate the expectations of interest by the following estimate:

$$
\begin{aligned}
\mathbf{E}[f_k(W_k)] &\approx \frac{1/N\sum_{i=1}^{N}f_k(W_k^{(i)})q_k(W_k^{(i)})}{1/N\sum_{i=1}^{N}q_k(W_k^{(i)})} \\
&= \sum_{i=1}^{N} f_k(W_k^{(i)})\tilde{q}_k(W_k^{(i)}) \tag{4}
\end{aligned}
$$

where the normalised importance ratios $\tilde{q}_k^{(i)}$ are given by:

$$\tilde{q}_k^{(i)} = \frac{q_k^{(i)}}{\sum_{j=1}^{N} q_k^{(j)}}$$

It is not difficult to show (de Freitas, Niranjan, Gee and Doucet 1998) that, if we assume $\mathbf{w}$ to be a hidden Markov process with initial density $\mathrm{p}(\mathbf{w}_0)$ and transition density $\mathrm{p}(\mathbf{w}_k|\mathbf{w}_{k-1})$, various recursive algorithms can be derived. One of these algorithms (HySIR), which we derive in (de Freitas, Niranjan, Gee and Doucet 1998), has been shown to perform well in neural network training. Here we extended the algorithm to deal with multiple noise levels. The pseudo-code for the HySIR algorithm with EKF updating is as follows[1]:

1. **INITIALISE NETWORK WEIGHTS** $(k = 0)$:

2. **For** $k = 1, \cdots, L$

    (a) **SAMPLING STAGE:**

        For $i = 1, \cdots, N$

- **Predict via the dynamics equation:**

$$\hat{\mathbf{w}}_{k+1}^{(i)} = \mathbf{w}_k^{(i)} + \mathbf{d}_k^{(i)}$$

where $\mathbf{d}_k^{(i)}$ is a sample from $p(\mathbf{d}_k)$ ($\mathcal{N}(0, Q_k)$ in our case).

- **Update samples with the EKF equations.**

$$
\begin{aligned}
\mathbf{w}_{k+1|k}^{(i)} &= \hat{\mathbf{w}}_{k+1}^{(i)} \\
P_{k+1|k}^{(i)} &= P_k^{T(i)} + Q^{*(i)} I_{mm} \\
K_{k+1}^{(i)} &= P_{k+1|k}^{(i)} G_{k+1}^{(i)} [R^* I_{oo} + G_{k+1}^{T(i)} P_{k+1|k}^{(i)} G_{k+1}^{(i)}]^{-1} \\
\hat{\mathbf{w}}_{k+1}^{(i)} &= \mathbf{w}_{k+1|K}^{(i)} + K_{k+1}^{(i)} (\mathbf{y}_{k+1} - \mathbf{g}(\mathbf{x}_{k+1}, \mathbf{w}_{k+1|k}^{(i)})) \\
\hat{P}_{k+1}^{(i)} &= P_{k+1|k}^{(i)} - K_{k+1}^{(i)} G_{k+1}^{T(i)} P_{k+1|k}^{(i)}
\end{aligned}
$$

- **Evaluate the importance ratios:**

$$q_{k+1}^{(i)} = q_k^{(i)} p(\mathbf{y}_{k+1} | \hat{\mathbf{w}}_{k+1}^{(i)}) = q_k^{(i)} \mathcal{N}(\mathbf{g}(\mathbf{x}_{k+1}, \hat{\mathbf{w}}_{k+1}^{(i)}), R_k)$$

- **Normalise the importance ratios:**

$$\tilde{q}_{k+1}^{(i)} = \frac{q_{k+1}^{(i)}}{\sum_{j=1}^{N} q_{k+1}^{(j)}}$$

    (b) **RESAMPLING STAGE:**

        For $i = 1, \cdots, N$

        If $N_{eff} \geq$ **Threshold:**

- $\mathbf{w}_{k+1}^{(i)} = \hat{\mathbf{w}}_{k+1}^{(i)}$
- $P_{k+1}^{(i)} = \hat{P}_{k+1}^{(i)}$
- $Q_{k+1}^{*(i)} = Q_{k+1}^{*(i)}$

        **Else**

- **Resample new index** $j$ **from the discrete set** $\{\hat{\mathbf{w}}_{k+1}^{(i)}, \tilde{q}_{k+1}^{(i)}\}$
- $\mathbf{w}_{k+1}^{(i)} = \hat{\mathbf{w}}_{k+1}^{(j)}$, $P_{k+1}^{(i)} = \hat{P}_{k+1}^{(j)}$ and $Q_{k+1}^{*(i)} = Q_{k+1}^{*(j)}$
- $q_{k+1}^{(i)} = \frac{1}{N}$

where $K_{k+1}$ is known as the Kalman gain matrix, $I_{mm}$ denotes the identity matrix of size $m \times m$, and $R^*$ and $\hat{Q}^*$ are two tuning parameters, whose roles are explained in detail in (de Freitas, Niranjan and Gee 1997). $G$ represents the Jacobian matrix and, strictly speaking, $P_k$ is an approximation to the covariance matrix of the network weights. The resampling stage is used to eliminate samples with low probability and multiply samples with high probability. Various authors have described efficient algorithms for accomplishing this task in $\mathcal{O}(N)$ operations (Pitt and Shephard 1997, Carpenter, Clifford and Fearnhead 1997, Doucet 1998).

# 4   EXPERIMENT

To assess the ability of the hybrid algorithm to estimate time-varying hidden parameters, we generated input-output data from a logistic function followed by a linear scaling and a displacement as shown in Figure 1. This simple model is equivalent to an MLP with one hidden neuron and an output linear neuron. We applied two Gaussian ($\mathcal{N}(0, 10)$) input sequences to the model and corrupted the weights and output values with Gaussian noise ($\mathcal{N}(0, 1 \times 10^{-3})$ and $\mathcal{N}(0, 1 \times 10^{-4})$ respectively). We then trained a second model with the same structure using the input-output

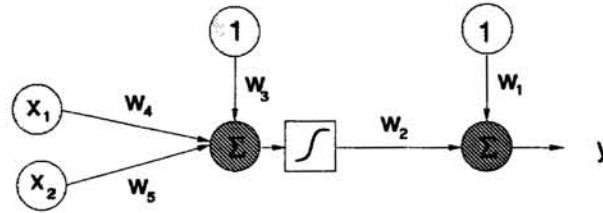

Figure 1: Logistic function with linear scaling and displacement used in the experiment. The weights were chosen as follows: $\mathbf{w}_1(k) = 1 + k/100$, $\mathbf{w}_2(k) = \sin(0.06k) - 2$, $\mathbf{w}_3(k) = 0.1$, $\mathbf{w}_4(k) = 1$, $\mathbf{w}_5(k) = -0.5$.

data generated by the first model. In so doing, we chose 100 sampling trajectories and set $R$ to 10, $Q$ to $1 \times 10^{-3} I_{55}$, the initial weights variance to 5, $P_0$ to $100 I_{55}$, $R^*$ to $1 \times 10^{-5}$. The process noise parameter $Q^*$ was set to three levels: $5 \times 10^{-3}$, $1 \times 10^{-3}$ and $1 \times 10^{-10}$, as shown in the plot of Figure 2 at time zero. In the training

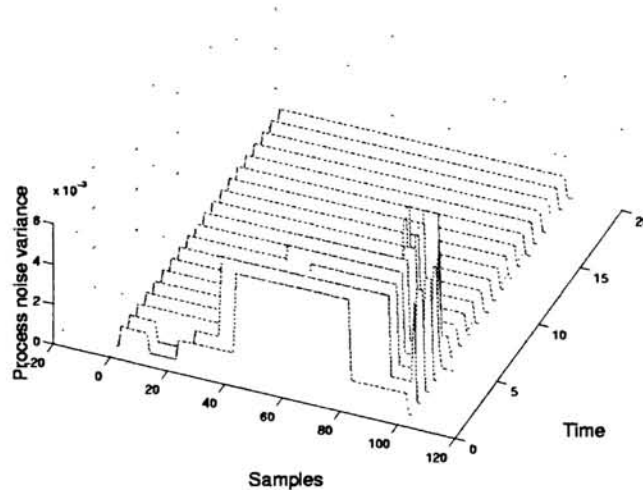

Figure 2: Noise level estimation with the HySIR algorithm.

phase, of 200 time steps, we allowed the model weights to vary with time. During this phase, the HySIR algorithm was used to track the input-output training data and estimate the latent model weights. In addition, we assumed three possible noise variance levels at the begining of the training session. After the 200-th time step, we fixed the values of the weights and generated another 200 input-output data test sets from the original model. The input test data was then fed to the trained model, using the weights values estimated at the 200-th time step. Subsequently,

the output prediction of the trained model was compared to the output data from the original model to assess the generalisation performance of the training process. As shown in Figure 2, the noise level of the trajectories converged to the true value $(1 \times 10^{-3})$. In addition, it was possible to track the network weights and obtain accurate output predictions as shown in Figures 3 and 4.

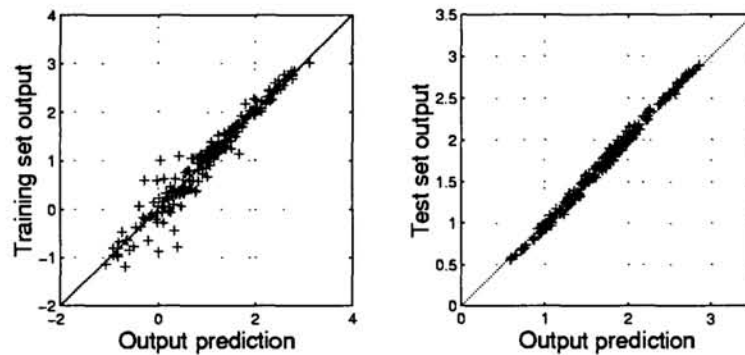

Figure 3: One step ahead predictions during the training phase (left) and stationary predictions in the test phase (right).

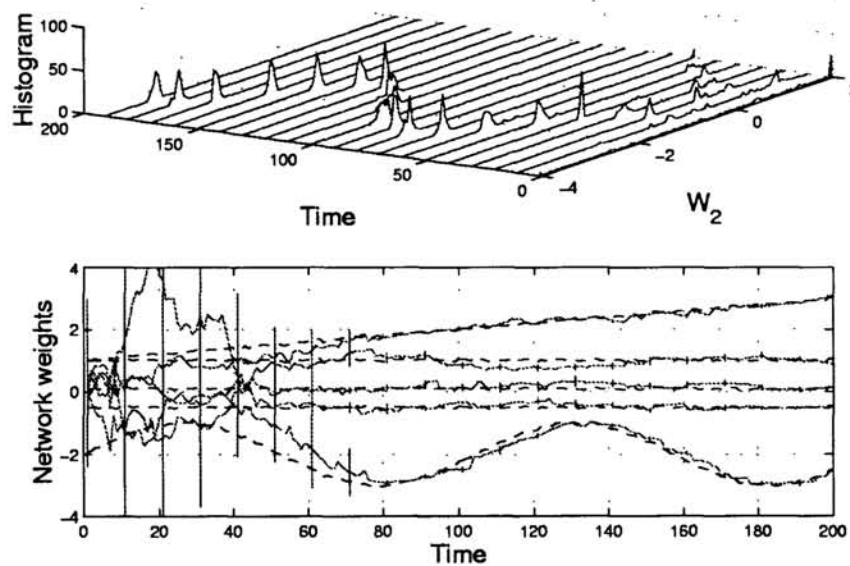

Figure 4: Weights tracking performance with the HySIR algorithm. As indicated by the histograms of $\mathbf{w}_2$, the algorithm performs a global search in parameter space.

## 5   CONCLUSIONS

In this paper, we have presented a sequential Monte Carlo approach for training neural networks in a Bayesian setting. In particular, we proposed an algorithm (HySIR) that makes use of both gradient and sampling information. HySIR can be interpreted as a Gaussian mixture filter, in that only a few sampling trajectories need to be employed. Yet, as the number of trajectories increases, the computational requirements increase only linearly. Therefore, the method is also suitable as a sampling strategy for approximating multi-modal distributions. Further avenues of research include the design of algorithms for adapting the noise covariances $R$ and $Q$, studying the effect of different noise models for the network weights and improving the computational efficiency of the algorithms.

## ACKNOWLEDGEMENTS

João FG de Freitas is financially supported by two University of the Witwatersrand Merit Scholarships, a Foundation for Research Development Scholarship (South Africa), an ORS award and a Trinity College External Studentship (Cambridge).

## Footnotes

[1]We have made available the software for the implementation of the HySIR algorithm at the following web-site: http://svr-www.eng.cam.ac.uk/~jfgf/software.html.

## References

Carpenter, J., Clifford, P. and Fearnhead, P. (1997). An improved particle filter for non-linear problems, *Technical report*, Department of Statistics, Oxford University, England. Available at http://www.stats.ox.ac.uk/~clifford/index.htm.

de Freitas, J. F. G., Niranjan, M. and Gee, A. H. (1997). Hierarchichal Bayesian-Kalman models for regularisation and ARD in sequential learning, *Technical Report CUED/F-INFENG/TR 307*, Cambridge University, http://svr-www.eng.cam.ac.uk/~jfgf.

de Freitas, J. F. G., Niranjan, M. and Gee, A. H. (1998). Regularisation in sequential learning algorithms, *in* M. I. Jordan, M. J. Kearns and S. A. Solla (eds), *Advances in Neural Information Processing Systems*, Vol. 10, MIT Press.

de Freitas, J. F. G., Niranjan, M., Gee, A. H. and Doucet, A. (1998). Sequential Monte Carlo methods for optimisation of neural network models, *Technical Report CUED/F-INFENG/TR 328*, Cambridge University, http://svr-www.eng.cam.ac.uk/~jfgf.

Doucet, A. (1998). On sequential simulation-based methods for Bayesian filtering, *Technical Report CUED/F-INFENG/TR 310*, Cambridge University. Available at http://www.stats.bris.ac.uk:81/MCMC/pages/list.html.

Gordon, N. J., Salmond, D. J. and Smith, A. F. M. (1993). Novel approach to nonlinear/non-Gaussian Bayesian state estimation, *IEE Proceedings-F* **140**(2): 107–113.

Pitt, M. K. and Shephard, N. (1997). Filtering via simulation: Auxiliary particle filters, *Technical report*, Department of Statistics, Imperial College of London, England. Available at http://www.nuff.ox.ac.uk/economics/papers.

Singhal, S. and Wu, L. (1988). Training multilayer perceptrons with the extended Kalman algorithm, *in* D. S. Touretzky (ed.), *Advances in Neural Information Processing Systems*, Vol. 1, San Mateo, CA, pp. 133–140.
